# Backpropagation Convergence Via Deterministic Nonmonotone Perturbed Minimization

**O. L. Mangasarian & M. V. Solodov**
Computer Sciences Department
University of Wisconsin
Madison, WI 53706
Email: olvi@cs.wisc.edu, solodov@cs.wisc.edu

## Abstract

The fundamental backpropagation (BP) algorithm for training artificial neural networks is cast as a deterministic nonmonotone perturbed gradient method . Under certain natural assumptions, such as the series of learning rates diverging while the series of their squares converging, it is established that every accumulation point of the online BP iterates is a stationary point of the BP error function. The results presented cover serial and parallel online BP, modified BP with a momentum term, and BP with weight decay.

## 1 INTRODUCTION

We regard training artificial neural networks as an unconstrained minimization problem

$$\min_{x \in \Re^n} f(x) := \sum_{j=1}^{N} f_j(x) \tag{1}$$

where $f_j : \Re^n \to \Re$, $j = 1, \ldots, N$ are continuously differentiable functions from the $n$-dimensional real space $\Re^n$ to the real numbers $\Re$. Each function $f_j$ represents the error associated with the $j$-th training example, and $N$ is the number of examples in the training set. The $n$-dimensional variable space here is that of the weights associated with the arcs of the neural network and the thresholds of the hidden and

output units. For an explicit description of $f(x)$ see (Mangasarian, 1993). We note that our convergence results are equally applicable to any other form of the error function, provided that it is smooth.

BP (Rumelhart,Hinton & Williams, 1986; Khanna, 1989) has long been successfully used by the artificial intelligence community for training artificial neural networks. Curiously, there seems to be no published *deterministic* convergence results for this method. The primary reason for this is the **non**monotonic nature of the process. Every iteration of online BP is a step in the direction of negative gradient of a partial error function associated with a single training example, e.g. $f_j(x)$ in (1). It is clear that there is no guarantee that such a step will decrease the full objective function $f(x)$, which is the sum of the errors for *all* the training examples. Therefore a single iteration of BP may, in fact, increase rather than decrease the objective function $f(x)$ we are trying to minimize. This difficulty makes convergence analysis of BP a challenging problem that has currently attracted interest of many researchers (Mangasarian & Solodov, 1994; Gaivoronski, 1994; Grippo, 1994; Luo & Tseng, 1994; White, 1989).

By using *stochastic* approximation ideas (Kashyap,Blaydon & Fu, 1970; Ermoliev & Wets, 1988), White (White, 1989) has shown that, under certain stochastic assumptions, the sequence of weights generated by BP either diverges or converges almost surely to a point that is a stationary point of the error function. More recently, Gaivoronski obtained stronger stochastic results (Gaivoronski, 1994). It is worth noting that even if the data is assumed to be deterministic, the best that stochastic analysis can do is to establish convergence of certain sequences with probability one. This means that convergence is not guaranteed. Indeed, there may exist some noise patterns for which the algorithm diverges, even though this event is claimed to be unlikely.

By contrast, our approach is purely **deterministic**. In particular, we show that online BP can be viewed as an ordinary perturbed nonmonotone gradient-type algorithm for unconstrained optimization (Section 3). We note in the passing, that the term *gradient descent* which is widely used in the backpropagation and neural networks literature is incorrect. From an optimization point of view, online BP is *not* a descent method, because there is no guaranteed decrease in the objective function at each step. We thus prefer to refer to it as a *nonmonotone perturbed gradient* algorithm.

We give a convergence result for a serial (Algorithm 2.1), a parallel (Algorithm 2.2) BP, a modified BP with a momentum term, and BP with weight decay. To the best of our knowledge, there is no published convergence analysis, either stochastic or deterministic, for the latter three versions of BP. The proposed parallel algorithm is an attempt to accelerate convergence of BP which is generally known to be relatively slow.

## 2   CONVERGENCE OF THE BACKPROPAGATION ALGORITHM AND ITS MODIFICATIONS

We now turn our attention to the classical BP algorithm for training feedforward artificial neural networks with one layer of hidden units (Rumelhart,Hinton &

Williams, 1986; Khanna, 1989). Throughout our analysis the number of hidden units is assumed to be fixed. The choice of network topology is a separate issue that is not addressed in this work. For some methods for choosing the number of hidden units see (Courrien, 1993; Arai, 1993).

We now summarize our notation.

$N$ : Nonnegative integer denoting number of examples in the training set.

$i = 1, 2, \ldots$ : Index number of major iterations (epochs) of BP. Each major iteration consists of going through the entire set of error functions $f_1(x), \ldots, f_N(x)$.

$j = 1, \ldots, N$ : Index of minor iterations. Each minor iteration $j$ consists of a step in the direction of the negative gradient $-\nabla f_{m(j)}(z^{i,j})$ and a momentum step. Here $m(j)$ is an element of the permuted set $\{1, \ldots, N\}$, and $z^{i,j}$ is defined immediately below. Note that if the training set is randomly permuted after every epoch, the map $m(\cdot)$ depends on the index $i$. For simplicity, we skip this dependence in our notation.

$x^i$ : Iterate in $\Re^n$ of major iteration (epoch) $i = 1, 2, \ldots.$

$z^{i,j}$ : Iterate in $\Re^n$ of minor iteration $j = 1, \ldots, N$, within major iteration $i = 1, 2, \ldots.$ Iterates $z^{i,j}$ can be thought of as elements of a matrix with $N$ columns and infinite number of rows, with row $i$ corresponding to the $i$-th epoch of BP.

By $\eta_i$ we shall denote the *learning rate* (the coefficient multiplying the gradient), and by $\alpha_i$ the *momentum rate* (the coefficient multiplying the momentum term). For simplicity we shall assume that the learning and momentum rates remain fixed within each major iteration. In a manner similar to that of conjugate gradients (Polyak, 1987) we reset the momentum term to zero periodically.

**Algorithm 2.1. *Serial Online BP with a Momentum Term.***
*Start with any $x^0 \in \Re^n$. Having $x^i$, stop if $\nabla f(x^i) = 0$, else compute $x^{i+1}$ as follows :*

$$z^{i,1} = x^i \tag{2}$$

$$z^{i,j+1} = z^{i,j} - \eta_i \nabla f_{m(j)}(z^{i,j}) + \alpha_i \Delta z^{i,j}, \quad j = 1, \ldots, N \tag{3}$$

$$x^{i+1} = z^{i,N+1} \tag{4}$$

*where*

$$\Delta z^{i,j} = \begin{cases} 0 & \text{if } j = 1 \\ z^{i,j} - z^{i,j-1} & \text{otherwise} \end{cases} \tag{5}$$

$$0 < \eta_i < 1, \quad 0 \leq \alpha_i < 1$$

**Remark 2.1.** Note that the stopping criterion of this algorithm is typically that used in first order optimization methods, and is not explicitly related to the ability of the neural network to generalize. However, since we are concerned with convergence properties of BP as a numerical algorithm, this stopping criterion is

justified. Another point related to the issue of generalization versus convergence is the following. Our analysis allows the use of a *weight decay* term in the objective function (Hinton, 1986; Weigend,Huberman & Rumelhart, 1990) which often yields a network with better generalization properties. In the latter case the minimization problem becomes

$$\min_{x \in \Re^n} f(x) := \sum_{j=1}^{N} f_j(x) + \lambda \|x\|^2 \tag{6}$$

where $\lambda$ is a small positive scaling factor.

**Remark 2.2.** The choice of $\alpha_i = 0$ reduces Algorithm 2.1 to the original BP without a momentum term.

**Remark 2.3.** We can easily handle the "mini-batch" methods (Møller, 1992) by merely redefining the meaning of the partial error function $f_j$ to represent the error associated with a subset of training examples. Thus "mini-batch" methods also fall within our framework.

We next present a parallel modification of BP. Suppose we have $k$ parallel processors, $k \geq 1$. We consider a partition of the set $\{1, \ldots, N\}$ into the subsets $J_l$, $l = 1, \ldots, k$, so that each example is assigned to at least one processor. Let $N_l$ be the cardinality of the corresponding set $J_l$. In the *parallel* BP each processor performs one (or more) cycles of serial BP on its set of training examples. Then a synchronization step is performed that consists of averaging the iterates computed by all the $k$ processors. From the mathematical point of view this is equivalent to each processor $l \in \{1, \ldots, k\}$ handling the partial error function $f^l(x)$ associated with the corresponding set of training examples $J_l$. In this setting we have

$$f^l(x) = \sum_{j \in J_l} f_j(x) , \quad f(x) = \sum_{l=1}^{k} f^l(x)$$

We note that in training a neural network it might be advantageous to assign some training examples to more than one parallel processor. We thus allow for the possibility of overlapping sets $J_l$.

The notation for Algorithm 2.2 is similar to that for Algorithm 2.1, except for the index $l$ that is used to label the partial error function and minor iterates associated with the $l$-th parallel processor. We now state the parallel BP with a momentum term.

**Algorithm 2.2. *Parallel Online BP with a Momentum Term.***
*Start with any $x^0 \in \Re^n$. Having $x^i$, stop if $x^{i+1} = x^i$, else compute $x^{i+1}$ as follows :*

*(i) Parallelization. For each parallel processor $l \in \{1, \ldots, k\}$ do*

$$z_l^{i,1} = x^i \tag{7}$$

$$z_l^{i,j+1} = z_l^{i,j} - \eta_i \nabla f^l_{m(j)}(z_l^{i,j}) + \alpha_i \Delta z_l^{i,j}, \quad j = 1, \ldots, N_l \tag{8}$$

*where*

$$\Delta z_l^{i,j} = \begin{cases} 0 & \text{if } j = 1 \\ z_l^{i,j} - z_l^{i,j-1} & \text{otherwise} \end{cases} \tag{9}$$

$$0 < \eta_i < 1, \quad 0 \le \alpha_i < 1$$

*(ii) Synchronization*

$$x^{i+1} = \frac{1}{k} \sum_{l=1}^{k} z_l^{i,N_l+1} \tag{10}$$

We give below in Table 1 a flowchart of this algorithm.

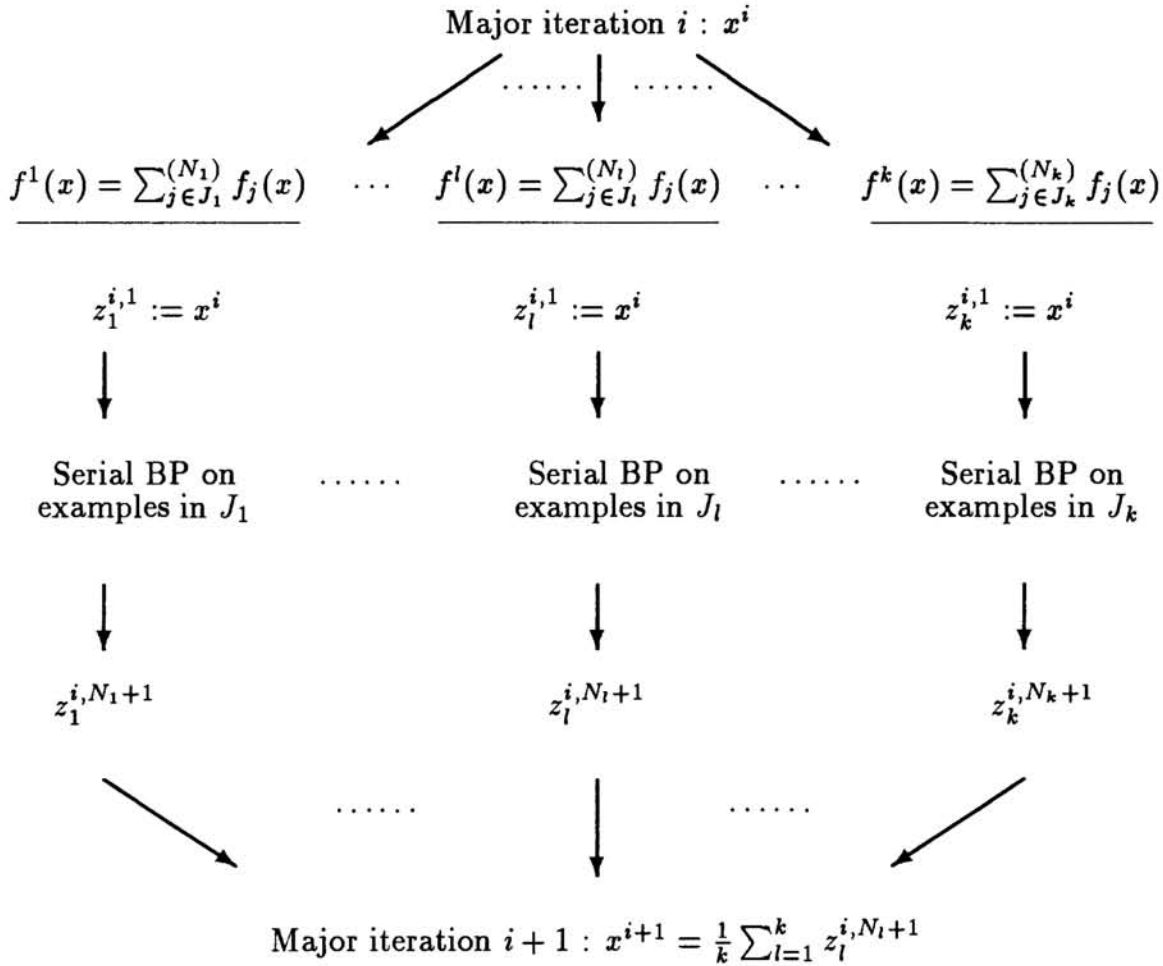

Major iteration $i$ : $x^i$

$$f^1(x) = \sum_{j \in J_1}^{(N_1)} f_j(x) \quad \cdots \quad f^l(x) = \sum_{j \in J_l}^{(N_l)} f_j(x) \quad \cdots \quad f^k(x) = \sum_{j \in J_k}^{(N_k)} f_j(x)$$

$z_1^{i,1} := x^i$      $z_l^{i,1} := x^i$      $z_k^{i,1} := x^i$

Serial BP on examples in $J_1$     Serial BP on examples in $J_l$     Serial BP on examples in $J_k$

$z_1^{i,N_1+1}$      $z_l^{i,N_l+1}$      $z_k^{i,N_k+1}$

Major iteration $i+1$ : $x^{i+1} = \frac{1}{k} \sum_{l=1}^{k} z_l^{i,N_l+1}$

**Table 1. Flowchart of the Parallel BP**

**Remark 2.4.** It is well known that ordinary backpropagation is a relatively slow algorithm. One appealing remedy is parallelization (Zhang,Mckenna,Mesirov & Waltz, 1990). The proposed Algorithm 2.2 is a possible step in that direction. Note that in Algorithm 2.2 all processors typically use the same program for their computations. Thus load balancing is easily achieved.

**Remark 2.5.** We wish to point out that synchronization strategies other than (10) are possible. For example, one may choose among the $k$ sets of weights and thresholds the one that best classifies the training data.

To the best of our knowledge there are no published deterministic convergence

proofs for either of Algorithms 2.1,2.2. Using new convergence analysis for a class of nonmonotone optimization methods with perturbations (Mangasarian & Solodov, 1994), we are able to derive deterministic convergence properties for online BP and its modifications. Once again we emphasize the equivalence of either of those methods to a deterministic nonmonotone perturbed gradient-type algorithm.

We now state our main convergence theorem. An important result used in the proof is given in the Mathematical Appendix. We refer interested readers to (Mangasarian & Solodov, 1994) for more details.

**Theorem 2.1.** *If the learning and momentum rates are chosen such that*

$$\sum_{i=0}^{\infty} \eta_i = \infty, \quad \sum_{i=0}^{\infty} \eta_i^2 < \infty, \quad \sum_{i=0}^{\infty} \alpha_i \eta_i < \infty, \tag{11}$$

*then for any sequence $\{x^i\}$ generated by any of the Algorithms 2.1 or 2.2, it follows that $\{f(x^i)\}$ converges, $\{\nabla f(x^i)\} \to 0$, and for each accumulation point $\bar{x}$ of the sequence $\{x^i\}$, $\nabla f(\bar{x}) = 0$.*

**Remark 2.6.** We note that conditions (11) imply that both the learning and momentum rates asymptotically tend to zero. These conditions are similar to those used in (White, 1989; Luo & Tseng, 1994) and seem to be the inevitable price paid for rigorous convergence. For practical purposes the learning rate can be fixed or adjusted to some small but finite number to obtain an approximate solution to the minimization problem. For state-of-the-art techniques of computing the learning rate see (le Cun, Simard & Pearlmutter, 1993).

**Remark 2.7.** We wish to point out that Theorem 2.1 covers BP with momentum and/or decay terms for which there is no published convergence analysis of any kind.

**Remark 2.8.** We note that the approach of perturbed minimization provides theoretical justification to the well known properties of robustness and recovery from damage for neural networks (Sejnowski & Rosenberg, 1987). In particular, this approach shows that the net should recover from any reasonably small perturbation.

**Remark 2.9.** Establishing convergence to a stationary point seems to be the best one can do for a first-order minimization method without any additional restrictive assumptions on the objective function. There have been some attempts to achieve global descent in training, see for example, (Cetin,Burdick & Barhen, 1993). However, convergence to global minima was not proven rigorously in the multidimensional case.

## 3   MATHEMATICAL APPENDIX: CONVERGENCE OF ALGORITHMS WITH PERTURBATIONS

In this section we state a new result that enables us to establish convergence properties of BP. The full proof is nontrivial and is given in (Mangasarian & Solodov, 1994).

**Theorem 3.1.  General Nonmonotonic Perturbed Gradient Convergence (subsumes BP convergence).**
*Suppose that $f(x)$ is bounded below and that $\nabla f(x)$ is bounded and Lipschitz continuous on the sequence $\{x^i\}$ defined below. Consider the following perturbed gradient algorithm. Start with any $x^0 \in \Re^n$. Having $x^i$, stop if $\nabla f(x^i) = 0$, else compute*

$$x^{i+1} = x^i + \eta_i d^i \tag{12}$$

*where*

$$d^i = -\nabla f(x^i) + e^i \tag{13}$$

*for some $e^i \in \Re^n$, $\eta_i \in \Re$, $\eta_i > 0$ and such that for some $\gamma > 0$*

$$\sum_{i=0}^{\infty} \eta_i = \infty, \;\; \sum_{i=0}^{\infty} \eta_i^2 < \infty, \;\; \sum_{i=0}^{\infty} \eta_i \|e^i\| < \infty, \;\; \|e^i\| \leq \gamma \;\; \forall i \tag{14}$$

*It follows that $\{f(x^i)\}$ converges, $\{\nabla f(x^i)\} \to 0$, and for each accumulation point $\bar{x}$ of the sequence $\{x^i\}$, $\nabla f(\bar{x}) = 0$. If, in addition, the number of stationary points of $f(x)$ is finite, then the sequence $\{x^i\}$ converges to a stationary point of $f(x)$.*

**Remark 3.1.** The error function of BP is nonnegative, and thus the boundedness condition on $f(x)$ is satisfied automatically. There are a number of ways to ensure that $f(x)$ has Lipschitz continuous and bounded gradient on $\{x^i\}$. In (Luo & Tseng, 1994) a simple projection onto a box is introduced which ensures that the iterates remain in the box. In (Grippo, 1994) a regularization term as in (6) is added to the error function so that the modified objective function has bounded level sets. We note that the latter provides a mathematical justification for weight decay (Hinton, 1986; Weigend,Huberman & Rumelhart, 1990). In either case the iterates remain in some compact set, and since $f(x)$ is an infinitely smooth function, its gradient is bounded and Lipschitz continuous on this set as desired.

### Acknowledgements

This material is based on research supported by Air Force Office of Scientific Research Grant F49620-94-1-0036 and National Science Foundation Grant CCR-9101801.

### References

M. Arai. (1993) Bounds on the number of hidden units in binary-valued three-layer neural networks. *Neural Networks*, 6:855–860.

B. C. Cetin, J. W. Burdick, and J. Barhen. (1993) Global descent replaces gradient descent to avoid local minima problem in learning with artificial neural networks. In *IEEE International Conference on Neural Networks, (San Francisco)*, volume 2, 836–842.

P. Courrien.(1993) Convergent generator of neural networks. *Neural Networks*, 6:835–844.

Yu. Ermoliev and R.J.-B. Wets (editors). (1988) *Numerical Techniques for Stochastic Optimization Problems*. Springer-Verlag, Berlin.

A.A. Gaivoronski. (1994) Convergence properties of backpropagation for neural networks via theory of stochastic gradient methods. Part 1. *Optimization Methods and Software*, 1994, to appear.

L. Grippo. (1994) A class of unconstrained minimization methods for neural network training. *Optimization Methods and Software*, 1994, to appear.

G. E. Hinton. (1986) Learning distributed representations of concepts. In *Proceedings of the Eighth Annual Conference of the Cognitive Science Society*, 1–12, Hillsdale. Erlbaum.

R. L. Kashyap, C. C. Blaydon and K. S. Fu. (1970) Applications of stochastic approximation methods. In J.M.Mendel and K.S. Fu, editors, *Adaptive, Learning, and Pattern Recognition Systems*. Academic Press.

T. Khanna. (1989) *Foundations of neural networks*. Addison–Wesley, New Jersey.

Y. le Cun, P.Y. Simard, and B. Pearlmutter. (1993) Automatic learning rate maximization by on-line estimation of the Hessian's eigenvectors. In C.L.Giles S.J.Hanson, J.D.Cowan, editor, *Advances in Neural Information Processing Systems 5*, 156–163, San Mateo, California, Morgan Kaufmann.

Z.-Q. Luo and P. Tseng. (1994) Analysis of an approximate gradient projection method with applications to the backpropagation algorithm. *Optimization Methods and Software*, 1994, to appear.

O.L. Mangasarian. (1993) Mathematical programming in neural networks. *ORSA Journal on Computing*, 5(4), 349–360.

O.L. Mangasarian and M.V. Solodov. (1994) Serial and parallel backpropagation convergence via nonmonotone perturbed minimization. *Optimization Methods and Software*, 1994, to appear. Proceedings of Symposium on Parallel Optimization 3, Madison July 7-9, 1993.

M.F. Møller. (1992) Supervised learning on large redundant training sets. In *Neural Networks for Signal Processing 2*. IEEE Press.

B.T. Polyak. (1987) *Introduction to Optimization*. Optimization Software, Inc., Publications Division, New York.

D.E. Rumelhart, G.E. Hinton, and R.J. Williams. (1986) Learning internal representations by error propagation. In D.E. Rumelhart and J.L. McClelland, editors, *Parallel Distributed Processing*, 318–362, Cambridge, Massachusetts. MIT Press.

T.J. Sejnowski and C.R. Rosenberg. (1987) Parallel networks that learn to pronounce english text. *Complex Systems*, 1:145–168.

A.S. Weigend, B.A. Huberman, and D.E. Rumelhart. (1990) Predicting the future:a connectionist approach. *International Journal of Neural Systems*, 1:193–209.

H. White. (1989) Some asymptotic results for learning in single hidden-layer feedforward network models. *Journal of the American Statistical Association*, 84(408):1003–1013.

X. Zhang, M. Mckenna, J. P. Mesirov, and D. L. Waltz. (1990) The backpropagation algorithm on grid and hypercube architectures. *Parallel Computing*, 14:317–327.